# FaceSync: A linear operator for measuring synchronization of video facial images and audio tracks

Malcolm Slaney[1]
Interval Research
malcolm@ieee.org

Michele Covell[2]
Interval Research
covell@ieee.org

## Abstract

FaceSync is an optimal linear algorithm that finds the degree of synchronization between the audio and image recordings of a human speaker. Using canonical correlation, it finds the best direction to combine all the audio and image data, projecting them onto a single axis. FaceSync uses Pearson's correlation to measure the degree of synchronization between the audio and image data. We derive the optimal linear transform to combine the audio and visual information and describe an implementation that avoids the numerical problems caused by computing the correlation matrices.

## 1 Motivation

In many applications, we want to know about the synchronization between an audio signal and the corresponding image data. In a teleconferencing system, we might want to know which of the several people imaged by a camera is heard by the microphones; then, we can direct the camera to the speaker. In post-production for a film, clean audio dialog is often dubbed over the video; we want to adjust the audio signal so that the lip-sync is perfect. When analyzing a film, we want to know when the person talking is in the shot, instead of off camera. When evaluating the quality of dubbed films, we can measure of how well the translated words and audio fit the actor's face.

This paper describes an algorithm, FaceSync, that measures the degree of synchronization between the video image of a face and the associated audio signal. We can do this task by synthesizing the talking face, using techniques such as Video Rewrite [1], and then comparing the synthesized video with the test video. That process, however, is expensive. Our solution finds a linear operator that, when applied to the audio and video signals, generates an audio–video-synchronization-error signal. The linear operator gathers information from throughout the image and thus allows us to do the computation inexpensively.

Hershey and Movellan [2] describe an approach based on measuring the mutual information between the audio signal and individual pixels in the video. The correlation between the audio signal, $x$, and one pixel in the image $y$, is given by Pearson's correlation, $r$. The mutual information between these two variables is given by $I(x,y) = -1/2 \ log(1-r^2)$. They create movies that show the regions of the video that have high correlation with the audio;

1. Currently at IBM Almaden Research, 650 Harry Road, San Jose, CA 95120.
2. Currently at YesVideo.com, 2192 Fortune Drive, San Jose, CA 95131.

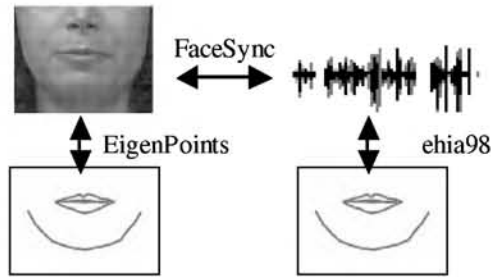

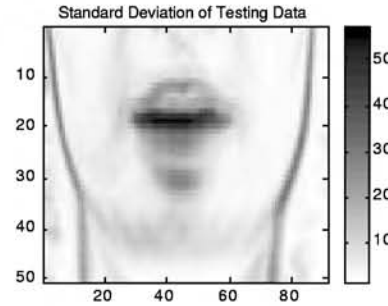

Figure 1: Connections between linear models relating audio, video and fiduciary points

Figure 2: Standard deviation of the aligned facial images used to create the canonical model.

from the correlation data, they estimate the centroid of the activity pattern and find the talking face. They make no claim of their algorithms ability to measure synchronization.

FaceSync is an optimal linear detector, equivalent to a Wiener filter [3], which combines the information from all the pixels to measure audio–video synchronization. We developed our approach based on two surprisingly simple algorithms in computer-vision and audio–visual speech synthesis: EigenPoints [4] and ATR's multilinear facial synthesizer [5]. The relationship of these two algorithms to each other and to our problem is shown in Figure 1.

EigenPoints [4] is an algorithm that finds a linear mapping between the brightness of a video signal and the location of fiduciary points on the face. At first, the validity of this mapping is not obvious; we might not expect the brightness of pixels on a face to covary linearly with $x$ and $y$ coordinates. It turns out, however, that the brightness of the image pixels, $i(x,y)$, and the location of fiduciary points such as the corner of the mouth, $p_i = (x_i, y_i)$, describe a function in a high-dimensional space. In the absence of occlusion, the combined brightness–fiduciary function is smoothly varying. Thus the derivatives are defined and a Taylor-series approximation is valid. The real surprise is that EigenPoints can find a linear approximation that describes the brightness–fiduciary space, and this linear approximation is valid over a useful range of brightness and control-point changes.

Similarly, Yehia, Rubin, and Vatikiotis-Bateson at ATR [5] have shown that it is possible to connect a specific model of speech, the line-spectral pairs or LSP, with the position of fiduciary points on the face. Their multilinear approximation yielded an average correlation of 0.91 between the true facial locations and those estimated from the audio data.

We derive a linear approximation to connect brightness to audio without the intermediate fiduciary points. Neither linear mapping is exact, so we had to determine whether the direct path between brightness and audio could be well approximated by a linear transform. We describe FaceSync in the next section.

Fisher and his colleagues [6] describe a more general approach that finds a non-linear mapping onto subspaces which maximize the mutual information. They report results using a single-layer perceptron for the non-linear mapping.

## 2 FaceSync Algorithm

FaceSync uses a face-recognition algorithm and canonical correlation to measure audio–visual synchrony. There are two steps: *training* or building the canonical correlation model, and *evaluating* the fit of the model to the data. In both steps we use face-recognition software to find faces and align them with a sample face image. In the training stage, canonical correlation finds a linear mapping that maximizes the cross-correlation between

two signals: the aligned face image and the audio signal. Finally, given new audio and video data, we use the linear mapping to rotate a new aligned face and the audio signal into a common space where we can evaluate their correlation as a function of time.

In both training and testing, we use a neural-network face-detection algorithm [7] to find portions of the image that contain a face. This approach uses a pyramid of images to search efficiently for pixels that look like faces. The software also allows the face to be tracked through a sequence of image and thus reduce the computational overhead, but we did not use this capability in our experiments. The output of Rowley's face-detection algorithm is a rectangle that encloses the position of a face. We use this information to align the image data prior to correlational analysis.

We investigated a number of ways to describe the audio signal. We looked at mel-frequency cepstral coefficients (MFCC) [8], linear-predictive coding (LPC) [8], line spectral frequencies (LSF) [9], spectrograms, and raw signal energy. For most calculations, we used MFCC analysis, because it is a favorite front-end for speech-recognition systems and, as do several of the other possibilities, it throws away the pitch information. This is useful because the pitch information affects the spectrogram in a non-linear manner and does not show up in the image data. For each form of audio analysis, we used a window size that was twice the frame interval (2 / 29.97 seconds,)

Canonical correlation analysis (CCA) uses jointly varying data from an input subspace $x_i$ and an output subspace $y_i$ to find canonic correlation matrices, $A_x$ and $A_y$. These matrices whiten the input and output data, as well as making the cross correlation diagonal and "maximally compact." Specifically, the whitened data matrices are

$$\eta = A_x^T(x - \bar{x}) \text{ and } \phi = A_y^T(y - \bar{y}), \tag{1}$$

and have the following properties:

$$E\{\eta\eta^T\} = I, \ E\{\phi\phi^T\} = I, \ E\{\phi\eta^T\} = \Sigma_K = diag\{\sigma_1, \sigma_2, ..., \sigma_L\}, \tag{2}$$

where $1 \geq \sigma_1 \geq \sigma_2 \geq ... > 0$ and $\sigma_{M+1} = ... = \sigma_L = 0$. In addition, for $i$ starting from 1 and then repeating up to $L$, $\sigma_i$ is the largest possible correlation between $\eta_i$ and $\varphi_i$ (where $\eta_i$ and $\varphi_i$ are the i[th] elements of $\eta$ and $\varphi$ respectively), given the norm and orthogonality constraints on $\eta$ and $\varphi$, expressed in equation 2. We refer to this property as maximal compaction, since the correlation is (recursively) maximally compacted into the leading elements of $\eta$ and $\varphi$.

We find the matrices $A_x$ and $A_y$ by whitening the input and output data:

$$x' = R_{xx}^{-1/2}(x - \bar{x}) \text{ and } y' = R_{yy}^{-1/2}(y - \bar{y}) \tag{3}$$

and then finding the left (U) and right (V) singular vectors of the cross-correlation matrix between the whitened data

$$K = R_{y'x'} = R_{yy}^{-1/2} R_{yx} R_{xx}^{-1/2} = U_K \Sigma_K V_K^T. \tag{4}$$

The SVD gives the same type of maximal compaction that we need for the cross correlation matrices, $A_x$ and $A_y$. Since the SVD is unique up to sign changes (and a couple of other degeneracies associated with repeated singular values), $A_x$ and $A_x$ must be:

$$A_x = R_{xx}^{-1/2} V_K \text{ and } A_y = R_{yy}^{-1/2} U_K. \tag{5}$$

We can verify this by calculating $E\{\varphi\eta^T\}$ using the definitions of $\varphi$ and $\eta$.

$$\varphi = A_y^T(y - \bar{y}) = (R_{yy}^{-1/2} U_K)^T (y - \bar{y}) = U_K^T R_{yy}^{-1/2}(y - \bar{y}), \tag{6}$$

$$\eta = A_x^T(x - \bar{x}) = (R_{xx}^{-1/2} V_K)^T (x - \bar{x}) = V_K^T R_{xx}^{-1/2}(x - \bar{x}), \tag{7}$$

then note

$$E\{\varphi\eta^T\} = U_K^T R_{yy}^{-1/2} E\{yx^T\} R_{xx}^{-1/2} V_K = U_K^T R_{yy}^{-1/2} R_{yx} R_{xx}^{-1/2} V_K \tag{8}$$

and then by using equation 4 (twice)

$$E\{\varphi\eta^T\} = U_K^T K V_K = U_K^T(U_K\Sigma_K V_K^T)V_K = (U_K^T U_K)\Sigma_K(V_K^T V_K) = \Sigma_K. \quad (9)$$

This derivation of canonical correlation uses correlation matrices. This introduces a well-known problem due to doubling the dynamic range of the analysis data. Instead, we formulate the estimation equations in terms of the components of the SVDs of the training data matrices. Specifically, we take the SVDs of the zero-mean input and output matrices:

$$[x_1 - \bar{x} \ldots x_N - \bar{x}] = \sqrt{N-1}\,U_x\Sigma_x V_x^T, [y_1 - \bar{y} \ldots y_N - \bar{y}] = \sqrt{N-1}\,U_y\Sigma_y V_y^T. \quad (10)$$

From these two decompositions, we can write the two correlation matrices as

$$R_{xx} = U_x\Sigma_x^2 U_x^T \qquad R_{xx}^{-1/2} = U_x\Sigma_x^{-1}U_x^T, \quad (11)$$

$$R_{yy} = U_y\Sigma_y^2 U_y^T \qquad R_{yy}^{-1/2} = U_y\Sigma_y^{-1}U_y^T, \quad (12)$$

and then write the cross-correlation matrix as

$$R_{yx} = U_y\Sigma_y V_y^T V_x^T \Sigma_x U_x^T. \quad (13)$$

Using these expressions for the correlation matrices, the $K$ matrix becomes

$$K = (U_y\Sigma^{-1}{}_y U_y^T)(U_y\Sigma_y V_y^T V_x\Sigma_x U_x^T)(U_x\Sigma_x^{-1}U_x^T) = U_y V_y^T V_x U_x^T. \quad (14)$$

Now let's look at the quantity $U_y^T K U_x$ in terms of its SVD

$$U_y^T K U_x = V_y^T V_x = (U_y^T U_K)\Sigma_K(V_K^T U_x) = U_{UKU}\Sigma_K V_{UKU}^T. \quad (15)$$

and, due to the uniqueness of the SVD, note

$$U_y^T U_k = U_{UKU} \text{ and } U_x^T V_K = V_{UKU}. \quad (16)$$

Now we can rewrite the equation for $A_x$ to remove the need for the squaring operation

$$A_x = R_{xx}^{-1/2}V_K = U_x\Sigma_x^{-1}(U_x^T V_K) = U_x\Sigma_x^{-1}V_{UKU} \quad (17)$$

and similarly for $A_y$

$$A_y = R_{yy}^{-1/2}U_K = U_y\Sigma_y^{-1}(U_y^T U_K) = U_y\Sigma_y^{-1}U_{UKU}. \quad (18)$$

Using these identities, we compute $A_x$ and $A_y$ using the following steps:

1)  Find the SVDs of the data matrices using the expressions in equation 10.

2)  Form a rotated version of the cross-correlation matrix $K$ and computes its SVD using equation 14.

3)  Compute the $A_x$ and $A_y$ matrixes using equations 17 and 18.

Given the linear mapping between audio data and the video images, as described by the $A_x$ and $A_y$ matrices, we measure the correlation between these two sets of data. For each candidate face in the image, we rotate the audio data by the first column of $A_x$, rotate the face image by the first column of $A_y$, and then compute Pearson's correlation of the rotated audio and video data. We use the absolute value of this correlation coefficient as a measure of audio–video synchronization.

## 3 Results

We evaluated the performance of the FaceSync algorithm using a number of tests. In the simplest tests we measured FaceSync's sensitivity to small temporal shifts between the audio and the video signals, evaluated our performance as a function of testing-window size and looked at different input representations. We also measured the effect of coarticulation.

To train the FaceSync system, we used 19 seconds of video. We used Rowley's face-detection software to find a rectangle bounding the face but we noticed a large amount (several

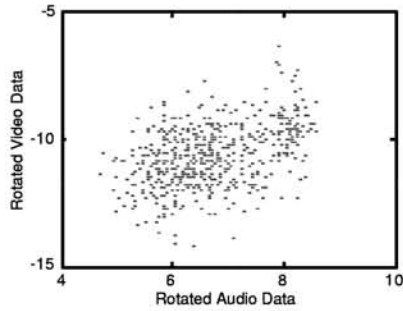

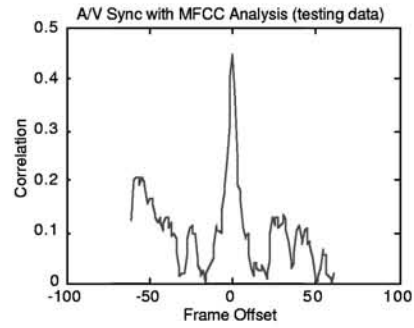

Figure 3: Optimum projections of the audio and video signals that maximize their cross-correlation.

Figure 4: Correlation of audio and video data as the audio data is shifted in time past the video. (29.97 frames/sec.)

pixels) of jitter in the estimated positions. Figure 2 shows the standard deviation of our aligned facial data. The standard deviation is high along the edges of the face, where small amounts of motion have a dramatic effect on the brightness, and around the mouth, where the image brightness changes with the spoken sounds.

Figure 3 shows the results of the canonical-correlation analysis for the 7 (distinct) seconds of audio and video that we used for testing. Canonical correlation has rotated the two multidimensional signals (audio and image) into the directions that are maximally correlated with each other. Note that the transformed audio and image signals are correlated.

We can evaluate the quality of these results by looking at the correlation of the two sets of data as the audio and image data are shifted relative to each other (such shifts are the kinds of errors that you would expect to see with bad lip sync.) An example of such a test is shown in Figure 4. Note that, after only a few frames of shift (about 100ms), the correlation between the audio and image data declined to close to zero.

We used the approach described by Hershey and Movellan to analyze which parts of the facial image correlate best with the audio data. In their work, they computed correlations over 16 frame intervals. Since we used aligned data, we could measure accurately the correlations over our entire 9 second test sequence. Our results are shown in Figure 5: Each pixel shows the correlation that we found using our data. This approach looks at each pixel individually and produces a maximum correlation near 0.45. Canonical correlation, which accumulates all the pixel information from all over the image, also produces a maximum correlation near 0.45, but by accumulating information from all over the image it allows us to measure sychronization without integrating over the full 9 seconds.

Figure 6 shows FaceSync's ability to measure audio–visual synchronization as we varied the testing-window size. For short windows (less than 1.3 seconds), we had insufficient data to measure the correlation accurately. For long windows (greater than 2.6 seconds), we had sufficient data to average and minimize the effect of errors, but as a result did not have high time resolution. As shown in Figure 5, there is a peak in the correlation near 0 frame offset; there are often, however, large noise peaks at other shifts. Between 1.3 and 2.6 seconds of video produces reliable results.

Different audio-analysis techniques provide different information to the FaceSync algorithm. Figure 7 shows the audio–video synchronization correlation, similar to Figure 3, for several different kinds of analysis. LPC and LSF produced identical narrow peaks; MFCC produced a slightly lower peak. Hershey used the power from the spectrogram in his algorithm to detect the visual motion. However, our result for spectrogram data is in the noise, indicating that a linear model can not use spectrogram data for fine-grain temporal measurements.

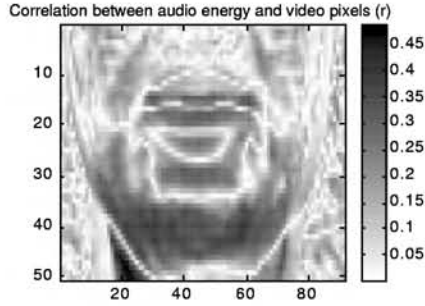

Figure 5: Correlation of each separate pixel and audio energy over the entire 9 second test sequence [2].

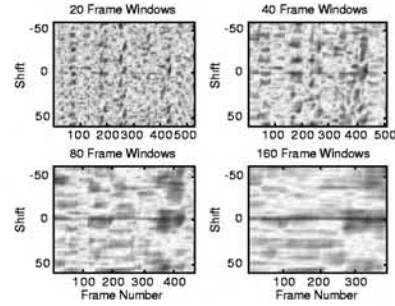

Figure 6: Performance of the FaceSync algorithm as a function of test window length. We would like to see a large peak (dark line) for all frames at zero shift.

We also looked at FaceSync's performance when we enhanced the video model with temporal context. Normally, we use one image frame and 67 ms of audio data as our input and output data. For this experiment, we stacked 13 images to form the input to the canonical-correlation algorithm Our performance did not vary as we added more visual context, probably indicating that a single image frame contained all of the information that the linear model was able to capture.

As the preceding experiment shows, we did not improve the performance by adding more image context. We can, however, use the FaceSync framework with extended visual context to learn something about co-articulation. Coarticulation is a well-known effect in speech; the audio and physical state of the articulators not only depends on the current phoneme, but also on the past history of the phonemic sequence and on the future sounds.

We let canonical correlation choose the most valuable data, across the range of shifted video images. Summing the squared weighting terms gives us an estimate of how much weight canonical correlation assigned to each shifted frame of data. Figure 8 shows that one video frame (30ms) before the current audio frame, and four video frames (120ms) after the current audio are affected by coarticulation. Interestingly, the zero-shift frame is not the one that shows the maximum importance. Instead, the frames just before and after are more heavily weighted.

## 4 Conclusions

We have described an algorithm, FaceSync, that builds an optimal *linear* model connecting the audio and video recordings of a person's speech. The model allows us to measure the degree of synchronization between the audio and video, so that we can, for example, determine who is speaking or to what degree the audio and video are sychronized.

While the goal of Hershey's process is not a temporal synchronization measurement, it is still interesting to compare the two approaches. Hershey's process does not take into account the mutual information between adjacent pixels; rather, it compares mutual information for individual pixels, then combines the results by calculating the centroid. In contrast, FaceSync asks what combination of audio and image data produces the best possible correlation, thus deriving a single optimal answer. Although the two algorithms both use Pearson's correlation to measure sychronization, FaceSync combines the pixels of the face and the audio information in an optimal detector.

The performance of the FaceSync algorithm is dependent on both training and testing data sizes. We did not test the quality of our models as we varied the training data. We do the training calculation only once using all the data we have. Most interesting applications of

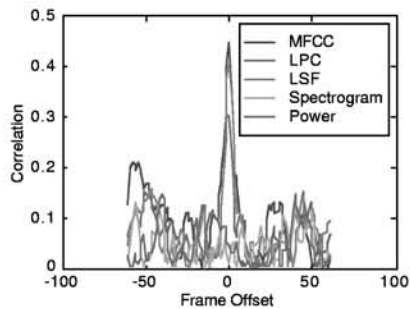
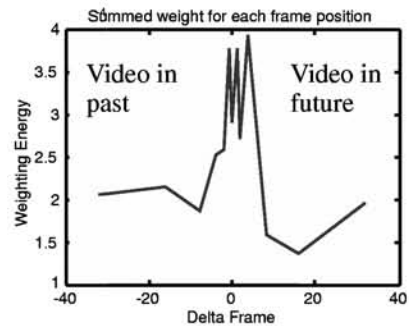

Figure 7: Performance of the FaceSync algorithm for different kinds of input representations.

Figure 8: Contributions of different frames to the optimum correlation with the audio frame

FaceSync depend on the testing data, and we would like to know how much data is necessary to make a decision.

In our FaceSync application, we have more dimensions (pixels in the image) than examples (video frames). Thus, our covariance matrices are singular, making their inversion — which we do as part of canonical correlation — problematic. We address the need for a pseudo-inverse, while avoiding the increased dynamic range of the covariance matrices, by using an SVD on the (unsquared) data matrices themselves (in place of an eigendecomposition of the covariance matrices).

We demonstrated high linear correlations between the audio and video signals, after we first found the optimal projection direction by using canonical correlation. We evaluated the FaceSync algorithm by measuring the correlation between the audio and video signals as we shift the audio data relative to the image data. MFCC, LPC, and LSF all produce sharp correlations as we shift the audio and images, whereas speech power and spectrograms produce no correlation peak at all.

## References

[1] C. Bregler, M. Covell, M. Slaney. "Video Rewrite: Driving visual speech with audio." *Proc. SIG-GRAPH 97*, Los Angeles, CA, pp. 353–360, August 1997.

[2] J. Hershey, J. R. Movellan. "Audio-Vision: Locating sounds via audio-visual synchrony." *Advances in Neural Information Processing Systems 12*, edited by S. A. Solla, T. K. Leen, K-R. Müller. MIT Press, Cambridge, MA (in press).

[3] L. L. Scharf, John K. Thomas. "Wiener filters in canonical coordinates for transform coding, filtering and quantizing." *IEEE Transactions on Signal Processing*, 46(3), pp. 647–654, March 1998.

[4] M. Covell, C. Bregler. "Eigenpoints." *Proc. Int. Conf. Image Processing,* Lausanne, Switzerland, Vol. 3, pp. 471–474, 1996.

[5] H. C. Yehia, P. E. Rubin, E. Vatikiotis-Bateson. "Quantitative association of vocal-tract and facial behavior," *Speech Communication*, 26, pp. 23–44, 1998.

[6] J. W. Fisher III, T. Darrell, W. T. Freeman, P. Viola. "Learning Joint Statistical Models for Audio-Visual Fusion and Segregation," This volume, 2001.

[7] H. A. Rowley, S. Baluja, and T. Kanade. "Neural network–based face detection." *IEEE Transactions on Pattern Analysis and Machine Intelligence*, 20(1), pp. 23–38, January 1998.

[8] L. Rabiner, B. Juang. *Fundamentals of Speech Recognition.* Prentice Hall, Englewood Cliffs, New Jersey, 1993.

[9] N. Sugamura, F. Itakura, "Speech analysis and synthesis methods developed at ECL in NTT—From LPC to LSP." *Speech Communications,* 4(2), June 1986.
